# A Model for Real-Time Computation in Generic Neural Microcircuits

**Wolfgang Maass** ,* **Thomas Natschläger**
Institute for Theoretical Computer Science
Technische Universitaet Graz, Austria
{maass, tnatschl}@igi.tu-graz.ac.at

**Henry Markram**
Brain Mind Institute
EPFL, Lausanne, Switzerland
henry.markram@epfl.ch

## Abstract

A key challenge for neural modeling is to explain how a continuous stream of multi-modal input from a rapidly changing environment can be processed by stereotypical recurrent circuits of integrate-and-fire neurons in real-time. We propose a new computational model that is based on principles of high dimensional dynamical systems in combination with statistical learning theory. It can be implemented on generic evolved or found recurrent circuitry.

## 1  Introduction

Diverse real-time information processing tasks are carried out by neural microcircuits in the cerebral cortex whose anatomical and physiological structure is quite similar in many brain areas and species. However a model that could explain the potentially universal computational capabilities of such recurrent circuits of neurons has been missing. Common models for the organization of computations, such as for example Turing machines or attractor neural networks, are not suitable since cortical microcircuits carry out computations on continuous streams of inputs. Often there is no time to wait until a computation has converged, the results are needed instantly ("anytime computing") or within a short time window ("real-time computing"). Furthermore biological data prove that cortical microcircuits can support several real-time computational tasks in parallel, a fact that is inconsistent with most modeling approaches. In addition the components of biological neural microcircuits, neurons and synapses, are highly diverse [1] and exhibit complex dynamical responses on several temporal scales. This makes them completely unsuitable as building blocks of computational models that require simple uniform components, such as virtually all models inspired by computer science or artificial neural nets. Finally computations in common computational models are partitioned into discrete steps, each of which require convergence to some stable internal state, whereas the dynamics of cortical microcircuits appears to be continuously changing. In this article we present a new conceptual framework for the organization of computations in cortical microcircuits that is not only compatible with all these constraints, but actually requires these biologically realistic features of neural computation. Furthermore like Turing machines this conceptual approach is supported by theoretical results that prove the universality of the computational model, but for the biologically more relevant case of real-time computing on continuous input streams.

*The work was partially supported by the Austrian Science Fond FWF, project #P15386.

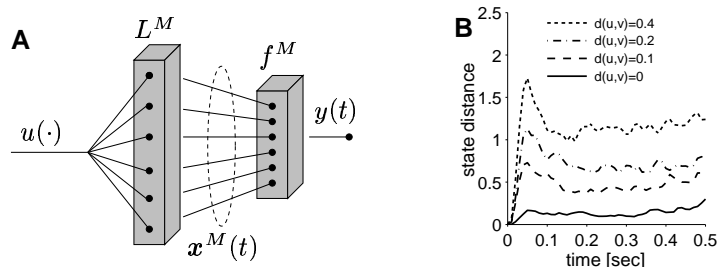

Figure 1: **A** Structure of a Liquid State Machine (LSM), here shown with just a single readout. **B** Separation property of a generic neural microcircuit. Plotted on the $y$-axis is the value of $\|x_u^M(t) - x_v^M(t)\|$, where $\|\cdot\|$ denotes the Euclidean norm, and $x_u^M(t)$, $x_v^M(t)$ denote the liquid states at time $t$ for Poisson spike trains $u$ and $v$ as inputs, averaged over many $u$ and $v$ with the same distance $d(u,v)$. $d(u,v)$ is defined as distance ($L_2$-norm) between low-pass filtered versions of $u$ and $v$.

## 2 A New Conceptual Framework for Real-Time Neural Computation

Our approach is based on the following observations. If one excites a sufficiently complex recurrent circuit (or other medium) with a continuous input stream $u(s)$, and looks at a later time $t > s$ at the current internal state $x(t)$ of the circuit, then $x(t)$ is likely to hold a substantial amount of information about recent inputs $u(s)$ (for the case of neural circuit models this was first demonstrated by [2]). We as human observers may not be able to understand the "code" by which this information about $u(s)$ is encoded in the current circuit state $x(t)$, but that is obviously not essential. Essential is whether a readout neuron that has to extract such information at time $t$ for a specific task can accomplish this. But this amounts to a classical pattern recognition problem, since the temporal dynamics of the input stream $u(s)$ has been transformed by the recurrent circuit into a high dimensional spatial pattern $x(t)$. A related approach for artificial neural nets was independently explored in [3].

In order to analyze the potential capabilities of this approach, we introduce the abstract model of a Liquid State Machine (LSM), see Fig. 1A. As the name indicates, this model has some weak resemblance to a finite state machine. But whereas the finite state set and the transition function of a finite state machine have to be custom designed for each particular computational task, a liquid state machine might be viewed as a universal finite state machine whose "liquid" high dimensional analog state $x(t)$ changes continuously over time. Furthermore if this analog state $x(t)$ is sufficiently high dimensional and its dynamics is sufficiently complex, then it has embedded in it the states and transition functions of many concrete finite state machines. Formally, an LSM $M$ consists of a filter $L^M$ (i.e. a function that maps input streams $u(\cdot)$ onto streams $x(\cdot)$, where $x(t)$ may depend not just on $u(t)$, but in a quite arbitrary nonlinear fashion also on previous inputs $u(s)$; in mathematical terminology this is written $x(t) = (L^M u)(t))$, and a (potentially memoryless) readout function $f^M$ that maps at any time $t$ the filter output $x(t)$ (i.e., the "liquid state") into some target output $y(t)$. Hence the LSM itself computes a filter that maps $u(\cdot)$ onto $y(\cdot)$.

In our application to neural microcircuits, the recurrently connected microcircuit could be viewed in a first approximation as an implementation of a general purpose filter $L^M$ (for example some unbiased analog memory), from which different readout neurons extract and recombine diverse components of the information contained in the input $u(\cdot)$. The liquid state $x(t)$ is that part of the internal circuit state at time $t$ that is accessible to readout neurons. An example where $u(\cdot)$ consists of 4 spike trains is shown in Fig. 2. The generic microcircuit model (270 neurons) was drawn from the distribution discussed in section 3.

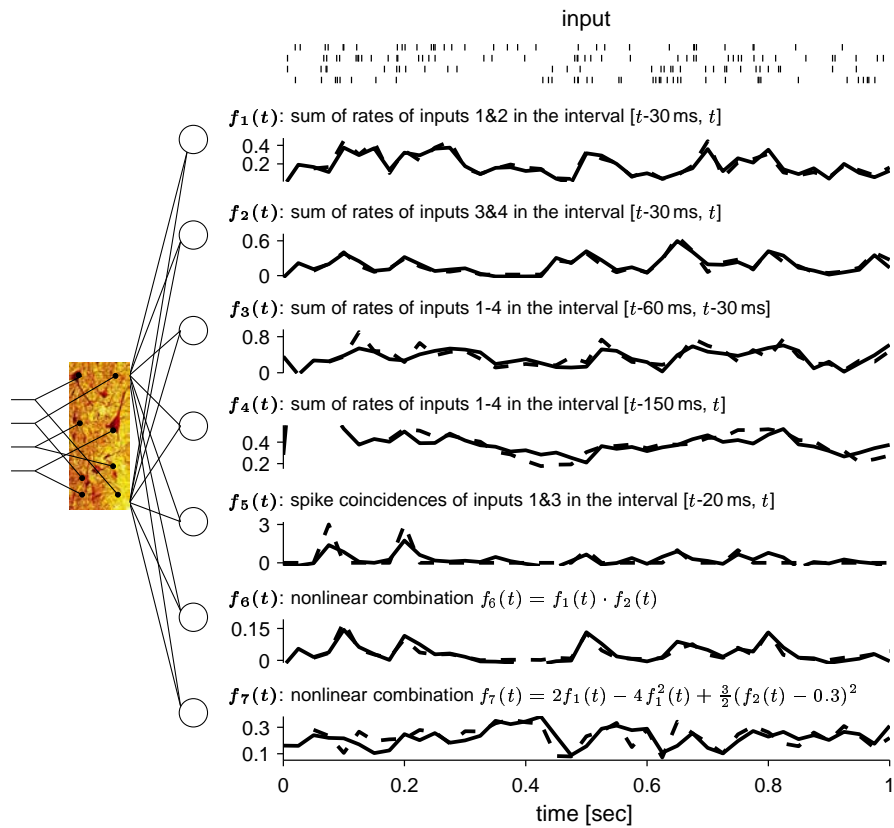

input

$f_1(t)$: sum of rates of inputs 1&2 in the interval $[t\text{-}30\,\text{ms}, t]$

0.4
0.2

$f_2(t)$: sum of rates of inputs 3&4 in the interval $[t\text{-}30\,\text{ms}, t]$

0.6
0

$f_3(t)$: sum of rates of inputs 1-4 in the interval $[t\text{-}60\,\text{ms}, t\text{-}30\,\text{ms}]$

0.8
0

$f_4(t)$: sum of rates of inputs 1-4 in the interval $[t\text{-}150\,\text{ms}, t]$

0.4
0.2

$f_5(t)$: spike coincidences of inputs 1&3 in the interval $[t\text{-}20\,\text{ms}, t]$

3
0

$f_6(t)$: nonlinear combination $f_6(t) = f_1(t) \cdot f_2(t)$

0.15
0

$f_7(t)$: nonlinear combination $f_7(t) = 2f_1(t) - 4f_1^2(t) + \frac{3}{2}(f_2(t) - 0.3)^2$

0.3
0.1

0     0.2     0.4     0.6     0.8     1
time [sec]

Figure 2: Multi-tasking in real-time. Input spike trains were randomly generated in such a way that at any time $t$ the input contained no information about preceding input more than 30 ms ago. Firing rates $r(t)$ were randomly drawn from the uniform distribution over [0 Hz, 80 Hz] every 30 ms, and input spike trains 1 and 2 were generated for the present 30 ms time segment as independent Poisson spike trains with this firing rate $r(t)$. This process was repeated (with independent drawings of $r(t)$ and Poission spike trains) for each 30 ms time segment. Spike trains 3 and 4 were generated in the same way, but with independent drawings of another firing rate $\tilde{r}(t)$ every 30 ms. The results shown in this figure are for test data, that were never before shown to the circuit. Below the 4 input spike trains the target (dashed curves) and actual outputs (solid curves) of 7 linear readout neurons are shown in real-time (on the same time axis). Targets were to output every 30 ms the actual firing rate (rates are normalized to a maximum rate of 80 Hz) of spike trains 1&2 during the preceding 30 ms ($f_1$), the firing rate of spike trains 3&4 ($f_2$), the sum of $f_1$ and $f_2$ in an earlier time interval $[t\text{-}60\,\text{ms}, t\text{-}30\,\text{ms}]$ ($f_3$) and during the interval $[t\text{-}150\,\text{ms},t]$ ($f_4$), spike coincidences between inputs 1&3 ($f_5(t)$ is defined as the number of spikes which are accompanied by a spike in the other spike train within 5 ms during the interval $[t\text{-}20\,\text{ms},t]$), a simple nonlinear combinations $f_6$ and a randomly chosen complex nonlinear combination $f_7$ of earlier described values. Since that all readouts were linear units, these nonlinear combinations are computed implicitly within the generic microcircuit model. Average correlation coefficients between targets and outputs for 200 test inputs of length 1 s for $f_1$ to $f_7$ were 0.91, 0.92, 0.79, 0.75, 0.68, 0.87, and 0.65.

In this case the 7 readout neurons $f_1$ to $f_7$ (modeled for simplicity just as linear units with a membrane time constant of 30 ms, applied to the spike trains from the neurons in the

circuit) were trained to extract completely different types of information from the input stream $u(\cdot)$, which require different integration times stretching from 30 to 150 ms. Since the readout neurons had a biologically realistic short time constant of just 30 ms, additional temporally integrated information had to be contained at any instance $t$ in the current firing state $x(t)$ of the recurrent circuit (its "liquid state"). In addition a large number of nonlinear combinations of this temporally integrated information are also "automatically" precomputed in the circuit, so that they can be pulled out by linear readouts. Whereas the information extracted by some of the readouts can be described in terms of commonly discussed schemes for "neural codes", this example demonstrates that it is hopeless to capture the dynamics or the information content of the primary engine of the neural computation, the liquid state of the neural circuit, in terms of simple coding schemes.

## 3 The Generic Neural Microcircuit Model

We used a randomly connected circuit consisting of leaky integrate-and-fire (I&F) neurons, 20% of which were randomly chosen to be inhibitory, as generic neural microcircuit model.[1] Parameters were chosen to fit data from microcircuits in rat somatosensory cortex (based on [1], [4] and unpublished data from the Markram Lab).[2] It turned out to be essential to keep the connectivity sparse, like in biological neural systems, in order to avoid chaotic effects.

In the case of a synaptic connection from $a$ to $b$ we modeled the synaptic dynamics according to the model proposed in [4], with the synaptic parameters $U$ (use), $D$ (time constant for depression), $F$ (time constant for facilitation) randomly chosen from Gaussian distributions that were based on empirically found data for such connections.[3] We have shown in [5] that without such synaptic dynamics the computational power of these microcircuit models decays significantly. For each simulation, the initial conditions of each I&F neuron, i.e. the membrane voltage at time $t = 0$, were drawn randomly (uniform distribution) from the interval [13.5 mV, 15.0 mV]. The "liquid state" $x(t)$ of the recurrent circuit consisting of $n$ neurons was modeled by an $n$-dimensional vector computed by applying a low pass filter with a time constant of 30 ms to the spike trains generated by the $n$ neurons in the recurrent microcicuit.

# 4 Towards a non-Turing Theory for Real-Time Neural Computation

Whereas the famous results of Turing have shown that one can construct Turing machines that are universal for digital sequential offline computing, we propose here an alternative computational theory that is more adequate for analyzing parallel real-time computing on analog input streams. Furthermore we present a theoretical result which implies that within this framework the computational units of the system can be quite arbitrary, provided that sufficiently diverse units are available (see the separation property and approximation property discussed below). It also is not necessary to *construct* circuits to achieve substantial computational power. Instead sufficiently large and complex "found" circuits (such as the generic circuit used as the main building block for Fig. 2) tend to have already large computational power, provided that the reservoir from which their units are chosen is sufficiently rich and diverse.

Consider a class $\mathcal{B}$ of basis filters $B$ (that may for example consist of the components that are available for building filters $L^M$ of neural LSMs, such as dynamic synapses). We say that this class $\mathcal{B}$ has the *point-wise separation property* if for any two input functions $u(\cdot), v(\cdot)$ with $u(s) \neq v(s)$ for some $s \leq t$ there exists some $B \in \mathcal{B}$ with $(Bu)(t) \neq (Bv)(t)$.[4] There exist completely different classes $\mathcal{B}$ of filters that satisfy this point-wise separation property: $\mathcal{B} = \{\text{all delay lines}\}$, $\mathcal{B} = \{\text{all linear filters}\}$, and biologically more relevant $\mathcal{B} = \{\text{models for dynamic synapses}\}$ (see [6]).

The complementary requirement that is demanded from the class $\mathcal{F}$ of functions from which the readout maps $f^M$ are to be picked is the well-known *universal approximation property*: for any continuous function $h$ and any closed and bounded domain one can approximate $h$ on this domain with any desired degree of precision by some $f \in \mathcal{F}$. An example for such a class is $\mathcal{F} = \{\text{feedforward sigmoidal neural nets}\}$. A rigorous mathematical theorem [5], states that for *any* class $\mathcal{B}$ of filters that satisfies the point-wise separation property and for *any* class $\mathcal{F}$ of functions that satisfies the universal approximation property one can approximate any given real-time computation on time-varying inputs with fading memory (and hence any biologically relevant real-time computation) by a LSM $M$ whose filter $L^M$ is composed of finitely many filters in $\mathcal{B}$, and whose readout map $f^M$ is chosen from the class $\mathcal{F}$. This theoretial result supports the following pragmatic procedure: In order to implement a given real-time computation with fading memory it suffices to take a filter $L$ whose dynamics is "sufficiently complex", and train a "sufficiently flexible" readout to assign for each time $t$ and state $x(t) = (Lu)(t)$ the target output $y(t)$. Actually, we found that if the neural microcircuit model is not too small, it usually suffices to use linear readouts. Thus the microcircuit automatically assumes "on the side" the computational role of a kernel for support vector machines.

For physical implementations of LSMs it makes more sense to study instead of the theoretically relevant point-wise separation property the following qualitative separation property as a test for the computational capability of a filter $L$: how different are the liquid states $x_u(t) = (Lu)(t)$ and $x_v(t) = (Lv)(t)$ for two different input histories $u(\cdot), v(\cdot)$. This is evaluated in Fig. 1B for the case where $u(\cdot), v(\cdot)$ are Poisson spike trains and $L$ is a generic neural microcircuit model. It turns out, that the difference between the liquid states scales roughly proportionally to the difference between the two input histories. This appears to be desirable from the practical point of view, since it implies that saliently different input histories can be distinguished more easily and in a more noise robust fashion by the readout. We propose to use such evaluation of the separation capability of neural microcircuits as a new standard test for their computational capabilities.

# 5  A Generic Neural Microcircuit on the Computational Test Stand

The theoretical results sketched in the preceding section can be interpreted as saying that there are no strong a priori limitations for the power of neural microcircuits for real-time computing with fading memory, provided they are sufficiently large and their components are sufficiently heterogeneous. In order to evaluate this somewhat surprising theoretical prediction, we use a well-studied computational benchmark task for which data have been made publicly available[5]: the speech recognition task considered in [7] and [8].

The dataset consists of 500 input files: the words "zero", "one", ..., "nine" are spoken by 5 different (female) speakers, 10 times by each speaker. The task was to construct a network of I&F neurons that could recognize each of the 10 spoken words $w$. Each of the 500 input files had been encoded in the form of 40 spike trains, with at most one spike per spike train [6] signaling onset, peak, or offset of activity in a particular frequency band. A network was presented in [8] that could solve this task with an error[7] of 0.15 for recognizing the pattern "one". No better result had been achieved by any competing networks constructed during a widely publicized internet competition [7]. The network constructed in [8] transformed the 40 input spike trains into linearly decaying input currents from 800 pools, each consisting of a "large set of closely similar unsynchronized neurons" [8]. Each of the 800 currents was delivered to a separate pair of neurons consisting of an excitatory "$\alpha$-neuron" and an inhibitory "$\beta$-neuron". To accomplish the particular recognition task some of the synapses between $\alpha$-neurons ($\beta$-neurons) are set to have equal weights, the others are set to zero. A particular achievement of this network (resulting from the smoothly and linearly decaying firing activity of the 800 pools of neurons) is that it is robust with regard to linear time-warping of the input spike pattern.

We tested our generic neural microcircuit model on the same task (in fact on exactly the same 500 input files). A randomly chosen subset of 300 input files was used for training, the other 200 for testing. The generic neural microcircuit model was drawn from the distribution described in section 3, hence from the same distribution as the circuit drawn for the completely different task discussed in Fig. 2, with randomly connected I&F neurons located on the integer points of a $15 \times 3 \times 3$ column. The synaptic weights of 10 linear readout neurons $f_w$ which received inputs from the 135 I&F neurons in the circuit were optimized (like for SVMs with linear kernels) to fire whenever the input encoded the spoken word $w$. Hence the whole circuit consisted of 145 I&F neurons, less than $1/30^{th}$ of the size of the network constructed in [8] for the same task[8]. Nevertheless the average error achieved after training by these randomly generated generic microcircuit models was 0.14 (measured in the same way, for the same word "one"), hence slightly better than that of the 30 times larger network custom designed for this task. The score given is the average for 50 randomly drawn generic microcircuit models.

The comparison of the two different approaches also provides a nice illustration of the

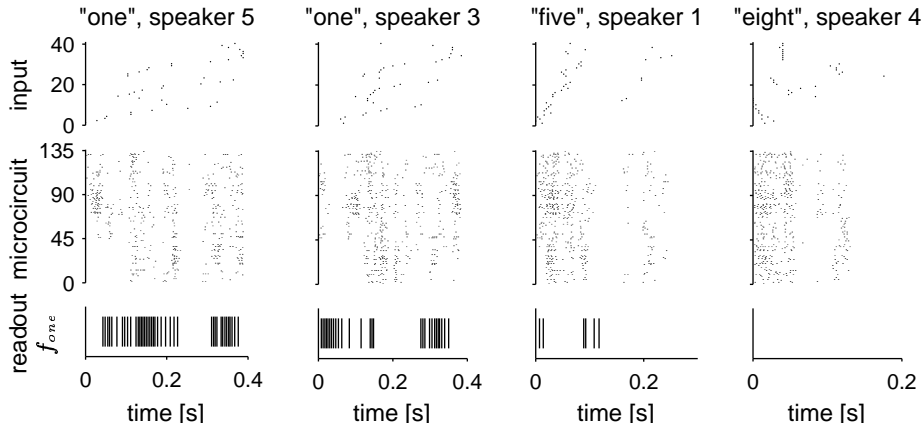

Figure 3: Application of our generic neural microcircuit model to the speech recognition from [8]. Top row: input spike patterns. Second row: spiking response of the 135 I&F neurons in the neural microcircuit model. Third row: output of an I&F neuron that was trained to fire as soon as possible when the word "one" was spoken, and as little as possible else.

difference between offline computing, real-time computing, and any-time computing. Whereas the network of [8] implements an algorithm that needs a few hundred ms of processing time between the end of the input pattern and the answer to the classification task (450 ms in the example of Fig. 2 in [8]), the readout neurons from the generic neural microcircuit were trained to provide their answer (through firing or non-firing) immediately when the input pattern ended. In fact, as illustrated in Fig. 3, one can even train the readout neurons quite successfully to provide provisional answers long before the input pattern has ended (thereby implementing an "anytime" algorithm). More precisely, each of the 10 linear readout neurons was trained to recognize the spoken word at any multiple of 20 ms while the word was spoken. An error score of 1.4 was achieved for this anytime speech recognition task.

We also compared the noise robustness of the generic microcircuit models with that of [8], which had been constructed to be robust with regard to linear time warping of the input pattern. Since no benchmark input data were available to calculate this noise robustness, we constructed such data by creating as templates 10 patterns consisting each of 40 randomly drawn Poisson spike trains at 4 Hz over 0.5 s. Noisy variations of these templates were created by first multiplying their time scale with a randomly drawn factor from $[1/3, 3]$) (thereby allowing for a 9 fold time warp), and subsequently dislocating each spike by an amount drawn independently from a Gaussian distribution with mean 0 and SD 32 ms. These spike patterns were given as inputs to the same generic neural microcircuit models consisting of 135 I&F neurons as discussed before. 10 linear readout neurons were trained (with 1000 randomly drawn training examples) to recognize which of the 10 templates had been used to generate a particular input. On 500 novel test examples (drawn from same distribution) they achieved an error of 0.09 (average performance of 30 randomly generated microcircuit models). As a consequence of achieving this noise robustness generically, rather then by a construction tailored to a specific type of noise, we found that the same generic microcircuit models are also robust with regard to *nonlinear* time warp of the input. For the case of nonlinear (sinusoidal) time warp [9] an average (50 microcircuits) error of 0.2

is achieved. This demonstrates that it is not necessary to build noise robustness explicitly into the circuit. A randomly generated microcircuit model has at least the same noise robustness as a circuit especially constructed to achieve that.

This test had implicitly demonstrated another point. Whereas the network of [8] was only able to classify spike patterns consisting of at most one spike per spike train, a generic neural microcircuit model can classify spike patterns without that restriction. It can for example also classify the original version of the speech data encoded into onsets, peaks, and offsets in various frequency bands, before all except the first events of each kind were artificially removed to fit the requirements of the network from [8].

The performance of the same generic neural microcircuit model on completely different computational tasks (recall of information from preceding input segments, movement prediction and estimation of the direction of movement of extended moving objects) turned out to be also quite remarkable, see [5], [9] and [10]. Hence this microcircuit model appears to have quite universal capabilities for real-time computing on time-varying inputs.

## 6 Discussion

We have presented a new conceptual framework for analyzing computations in generic neural microcircuit models that satisfies the biological constraints listed in section 1. Thus for the first time one can now take computer models of neural microcircuits, that can be as realistic as one wants to, and use them not just for demonstrating dynamic effects such as synchronization or oscillations, but to really carry out demanding computations with these models. Furthermore our new conceptual framework for analyzing computations in neural circuits not only provides theoretical support for their seemingly universal capabilities for real-time computing, but also throws new light on key concepts such as neural coding. Finally, since in contrast to virtually all computational models the generic neural microcircuit models that we consider have no preferred direction of information processing, they offer an ideal platform for investigating the interaction of bottom-up and top-down processing of information in neural systems.

## Footnotes

[1]The software used to simulate the model is available via www.lsm.tugraz.at .

[2]*Neuron parameters*: membrane time constant 30 ms, absolute refractory period 3 ms (excitatory neurons), 2 ms (inhibitory neurons), threshold 15 mV (for a resting membrane potential assumed to be 0), reset voltage 13.5 mV, constant nonspecific background current $I_b = 13.5$ nA, input resistance 1 MΩ. *Connectivity structure*: The probability of a synaptic connection from neuron $a$ to neuron $b$ (as well as that of a synaptic connection from neuron $b$ to neuron $a$) was defined as $C \cdot \exp(-D^2(a,b)/\lambda^2)$, where $\lambda$ is a parameter which controls both the average number of connections and the average distance between neurons that are synaptically connected (we set $\lambda = 2$, see [5] for details). We assumed that the neurons were located on the integer points of a 3 dimensional grid in space, where $D(a,b)$ is the Euclidean distance between neurons $a$ and $b$. Depending on whether $a$ and $b$ were excitatory ($E$) or inhibitory ($I$), the value of $C$ was 0.3 ($EE$), 0.2 ($EI$), 0.4 ($IE$), 0.1 ($II$).

[3]Depending on whether $a$ and $b$ were excitatory ($E$) or inhibitory ($I$), the mean values of these three parameters (with $D$,$F$ expressed in seconds, s) were chosen to be .5, 1.1, .05 ($EE$), .05, .125, 1.2 ($EI$), .25, .7, .02 ($IE$), .32, .144, .06 ($II$). The SD of each parameter was chosen to be 50% of its mean. The mean of the scaling parameter $A$ (in nA) was chosen to be 30 (EE), 60 (EI), -19 (IE), -19 (II). In the case of input synapses the parameter $A$ had a value of 18 nA if projecting onto a excitatory neuron and 9 nA if projecting onto an inhibitory neuron. The SD of the $A$ parameter was chosen to be 100% of its mean and was drawn from a gamma distribution. The postsynaptic current was modeled as an exponential decay $\exp(-t/\tau_s)$ with $\tau_s = 3$ ms ($\tau_s = 6$ ms) for excitatory (inhibitory) synapses. The transmission delays between liquid neurons were chosen uniformly to be 1.5 ms ($EE$), and 0.8 ms for the other connections.

[4]Note that it is *not* required that there exists a single $B \in \mathcal{B}$ which achieves this separation for any two different input histories $u(\cdot), v(\cdot)$.

[5] http://moment.princeton.edu/ mus/Organism/Competition/digits_data.html

[6] The network constructed in [8] required that each spike train contained at most one spike.

[7] The error (or "recognition score") $S$ for a particular word $w$ was defined in [8] by $S = \frac{N_{fp}}{N_{cp}} + \frac{N_{fn}}{N_{cn}}$, where $N_{fp}$ ($N_{cp}$) is the number of false (correct) positives and $N_{fn}$ and $N_{cn}$ are the numbers of false and correct negatives. We use the same definition of error to facilitate comparison of results. The recognition scores of the network constructed in [8] and of competing networks of other researchers can be found at http://moment.princeton.edu/m̃us/Organism/Docs/winners.html. For the competition the networks were allowed to be constructed especially for their task, but only one single pattern for each word could be used for setting the synaptic weights. Since our microcircuit models were not prepared for this task, they had to be trained with substantially more examples.

[8] If one assumes that each of the 800 "large" pools of neurons in that network would consist of just 5 neurons, it contains together with the $\alpha$ and $\beta$-neurons 5600 neurons.

[9] A spike at time $t$ was transformed into a spike at time $t' = g(t) := B + K \cdot (t + 1/(2\pi f) \cdot \sin(2\pi ft + \varphi))$ with $f = 2$ Hz, $K$ randomly drawn from [0.5,2], $\varphi$ randomly drawn from $[0, 2\pi]$ and $B$ chosen such that $g(0) = 0$.

## References

[1] A. Gupta, Y. Wang, and H. Markram. Organizing principles for a diversity of GABAergic interneurons and synapses in the neocortex. *Science*, 287:273–278, 2000.

[2] D. V. Buonomano and M. M. Merzenich. Temporal information transformed into a spatial code by a neural network with realistic properties. *Science*, 267:1028–1030, Feb. 1995 1995.

[3] H. Jaeger. The "echo state" approach to analysing and training recurrent neural networks. German National Research Center for Information Technology, Report 148, 2001.

[4] H. Markram, Y. Wang, and M. Tsodyks. Differential signaling via the same axon of neocortical pyramidal neurons. *Proc. Natl. Acad. Sci.*, 95:5323–5328, 1998.

[5] W. Maass, T. Natschläger, and H. Markram. Real-time computing without stable states: A new framework for neural computation based on perturbations. *Neur. Comp.*, 14:2531–2560, 2002.

[6] W. Maass and E. D. Sontag. Neural systems as nonlinear filters. *Neur. Comp.*, 12:1743–1772, 2000.

[7] J. J. Hopfield and C. D. Brody. What is a moment? "cortical" sensory integration over a brief interval. *Proc. Natl. Acad. Sci. USA*, 97(25):13919–13924, 2000.

[8] J. J. Hopfield and C. D. Brody. What is a moment? transient synchrony as a collective mechanism for spatiotemporal integration. *Proc. Natl. Acad. Sci. USA*, 98(3):1282–1287, 2001.

[9] W. Maass, R. A. Legenstein, and H. Markram. A new approach towards vision suggested by biologically realistic neural microcircuit models. In H. H. Buelthoff, S. W. Lee, T. A. Poggio, and C. Wallraven, editors, *Proc. of the 2nd International Workshop on Biologically Motivated Computer Vision 2002*, volume 2525 of *LNCS*, pages 282–293. Springer, 2002.

[10] W. Maass, T. Natschläger, and H. Markram. Computational models for generic cortical microcircuits. In J. Feng, editor, *Computational Neuroscience: A Comprehensive Approach*. CRC-Press, 2002. to appear.
